# Greedy Layer-Wise Training of Deep Networks

**Yoshua Bengio, Pascal Lamblin, Dan Popovici, Hugo Larochelle**
Université de Montréal
Montréal, Québec
{bengioy,lamblinp,popovicd,larocheh}@iro.umontreal.ca

## Abstract

Complexity theory of circuits strongly suggests that deep architectures can be much more efficient (sometimes exponentially) than shallow architectures, in terms of computational elements required to represent some functions. Deep multi-layer neural networks have many levels of non-linearities allowing them to compactly represent highly non-linear and highly-varying functions. However, until recently it was not clear how to train such deep networks, since gradient-based optimization starting from random initialization appears to often get stuck in poor solutions. Hinton et al. recently introduced a greedy layer-wise unsupervised learning algorithm for Deep Belief Networks (DBN), a generative model with many layers of hidden causal variables. In the context of the above optimization problem, we study this algorithm empirically and explore variants to better understand its success and extend it to cases where the inputs are continuous or where the structure of the input distribution is not revealing enough about the variable to be predicted in a supervised task. Our experiments also confirm the hypothesis that the greedy layer-wise unsupervised training strategy mostly helps the optimization, by initializing weights in a region near a good local minimum, giving rise to internal distributed representations that are high-level abstractions of the input, bringing better generalization.

## 1 Introduction

Recent analyses (Bengio, Delalleau, & Le Roux, 2006; Bengio & Le Cun, 2007) of modern non-parametric machine learning algorithms that are kernel machines, such as Support Vector Machines (SVMs), graph-based manifold and semi-supervised learning algorithms suggest fundamental limitations of some learning algorithms. The problem is clear in kernel-based approaches when the kernel is *"local"* (e.g., the Gaussian kernel), i.e., $K(x, y)$ converges to a constant when $||x - y||$ increases. These analyses point to the difficulty of learning *"highly-varying functions"*, i.e., functions that have a large number of *"variations"* in the domain of interest, e.g., they would require a large number of pieces to be well represented by a piecewise-linear approximation. Since the number of pieces can be made to grow exponentially with the number of factors of variations in the input, this is connected with the well-known curse of dimensionality for classical non-parametric learning algorithms (for regression, classification and density estimation). If the shapes of all these pieces are unrelated, one needs enough examples for each piece in order to generalize properly. However, if these shapes are related and can be predicted from each other, *"non-local" learning algorithms* have the potential to generalize to pieces not covered by the training set. Such ability would seem necessary for learning in complex domains such as Artificial Intelligence tasks (e.g., related to vision, language, speech, robotics).

Kernel machines (not only those with a local kernel) have a **shallow architecture**, i.e., only two levels of data-dependent computational elements. This is also true of feedforward neural networks with a single hidden layer (which can become SVMs when the number of hidden units becomes large (Bengio, Le Roux, Vincent, Delalleau, & Marcotte, 2006)). A serious problem with shallow architectures is that they can be very inefficient in terms of the number of computational units (e.g., bases, hidden units), and thus in terms of required examples (Bengio & Le Cun, 2007). One way to represent a highly-varying function compactly (with few parameters) is through the composition of many non-linearities, i.e., with a **deep architecture**. For example, the parity function with $d$ inputs requires $O(2^d)$ examples and parameters to be represented by a Gaussian SVM (Bengio et al., 2006), $O(d^2)$ parameters for a one-hidden-layer neural network, $O(d)$ parameters and units for a multi-layer network with $O(\log_2 d)$ layers, and $O(1)$ parameters with a recurrent neural network. More generally,

boolean functions (such as the function that computes the multiplication of two numbers from their $d$-bit representation) expressible by $O(\log d)$ layers of combinatorial logic with $O(d)$ elements in each layer may require $O(2^d)$ elements when expressed with only 2 layers (Utgoff & Stracuzzi, 2002; Bengio & Le Cun, 2007). When the representation of a concept requires an exponential number of elements, e.g., with a shallow circuit, the number of training examples required to learn the concept may also be impractical. Formal analyses of the computational complexity of shallow circuits can be found in (Hastad, 1987) or (Allender, 1996). They point in the same direction: shallow circuits are much less expressive than deep ones.

However, until recently, it was believed too difficult to train deep multi-layer neural networks. Empirically, deep networks were generally found to be not better, and often worse, than neural networks with one or two hidden layers (Tesauro, 1992). As this is a negative result, it has not been much reported in the machine learning literature. A reasonable explanation is that gradient-based optimization starting from random initialization may get stuck near poor solutions. An approach that has been explored with some success in the past is based on *constructively* adding layers. This was previously done using a supervised criterion at each stage (Fahlman & Lebiere, 1990; Lengellé & Denoeux, 1996). Hinton, Osindero, and Teh (2006) recently introduced a greedy layer-wise *unsupervised* learning algorithm for Deep Belief Networks (DBN), a generative model with many layers of hidden causal variables. The training strategy for such networks may hold great promise as a principle to help address the problem of training deep networks. Upper layers of a DBN are supposed to represent more "abstract" concepts that explain the input observation $x$, whereas lower layers extract "low-level features" from $x$. They learn simpler concepts first, and build on them to learn more abstract concepts. This strategy, studied in detail here, has not yet been much exploited in machine learning. We hypothesize that three aspects of this strategy are particularly important: first, pre-training one layer at a time in a greedy way; second, using unsupervised learning at each layer in order to preserve information from the input; and finally, fine-tuning the whole network with respect to the ultimate criterion of interest.

We first extend DBNs and their component layers, Restricted Boltzmann Machines (RBM), so that they can more naturally handle continuous values in input. Second, we perform experiments to better understand the advantage brought by the greedy layer-wise unsupervised learning. The basic question to answer is whether or not this approach helps to solve a difficult optimization problem. In DBNs, RBMs are used as building blocks, but applying this same strategy using auto-encoders yielded similar results. Finally, we discuss a problem that occurs with the layer-wise greedy unsupervised procedure when the input distribution is not revealing enough of the conditional distribution of the target variable given the input variable. We evaluate a simple and successful solution to this problem.

## 2 Deep Belief Nets

Let $x$ be the input, and $\mathbf{g}^i$ the hidden variables at layer $i$, with joint distribution

$$P(x, \mathbf{g}^1, \mathbf{g}^2, \dots, \mathbf{g}^\ell) = P(x|\mathbf{g}^1)P(\mathbf{g}^1|\mathbf{g}^2) \cdots P(\mathbf{g}^{\ell-2}|\mathbf{g}^{\ell-1})P(\mathbf{g}^{\ell-1}, \mathbf{g}^\ell),$$

where all the conditional layers $P(\mathbf{g}^i|\mathbf{g}^{i+1})$ are factorized conditional distributions for which computation of probability and sampling are easy. In Hinton et al. (2006) one considers the hidden layer $\mathbf{g}^i$ a binary random vector with $n^i$ elements $\mathbf{g}^i_j$:

$$P(\mathbf{g}^i|\mathbf{g}^{i+1}) = \prod_{j=1}^{n^i} P(\mathbf{g}^i_j|\mathbf{g}^{i+1}) \ \text{ with } \ P(\mathbf{g}^i_j = 1|\mathbf{g}^{i+1}) = \text{sigm}(b^i_j + \sum_{k=1}^{n^{i+1}} W^i_{kj}\mathbf{g}^{i+1}_k) \quad (1)$$

where $\text{sigm}(t) = 1/(1 + e^{-t})$, the $b^i_j$ are *biases* for unit $j$ of layer $i$, and $W^i$ is the *weight matrix* for layer $i$. If we denote $\mathbf{g}^0 = x$, the generative model for the first layer $P(x|\mathbf{g}^1)$ also follows (1).

### 2.1 Restricted Boltzmann machines

The top-level prior $P(\mathbf{g}^{\ell-1}, \mathbf{g}^\ell)$ is a **Restricted Boltzmann Machine** (RBM) between layer $\ell - 1$ and layer $\ell$. To lighten notation, consider a generic RBM with input layer activations $\mathbf{v}$ (for *visible* units) and hidden layer activations $\mathbf{h}$ (for *hidden* units). It has the following joint distribution: $P(\mathbf{v}, \mathbf{h}) = \frac{1}{Z}e^{\mathbf{h}'W\mathbf{v}+b'\mathbf{v}+c'\mathbf{h}}$, where $Z$ is the normalization constant for this distribution, $b$ is the vector of biases for visible units, $c$ is the vector of biases for the hidden units, and $W$ is the weight matrix for the layer. Minus the argument of the exponential is called the **energy function**,

$$\text{energy}(\mathbf{v}, \mathbf{h}) = -\mathbf{h}'W\mathbf{v} - b'\mathbf{v} - c'\mathbf{h}. \quad (2)$$

We denote the RBM parameters together with $\theta = (W, b, c)$. We denote $Q(\mathbf{h}|\mathbf{v})$ and $P(\mathbf{v}|\mathbf{h})$ the layer-to-layer conditional distributions associated with the above RBM joint distribution.

The layer-to-layer conditionals associated with the RBM factorize like in (1) and give rise to $P(\mathbf{v}_k = 1|\mathbf{h}) = \text{sigm}(b_k + \sum_j W_{jk}\mathbf{h}_j)$ and $Q(\mathbf{h}_j = 1|\mathbf{v}) = \text{sigm}(c_j + \sum_k W_{jk}\mathbf{v}_k)$.

## 2.2 Gibbs Markov chain and log-likelihood gradient in an RBM

To obtain an estimator of the gradient on the log-likelihood of an RBM, we consider a Gibbs Markov chain on the (visible units, hidden units) pair of variables. Gibbs sampling from an RBM proceeds by sampling $\mathbf{h}$ given $\mathbf{v}$, then $\mathbf{v}$ given $\mathbf{h}$, etc. Denote $\mathbf{v}_t$ for the $t$-th $\mathbf{v}$ sample from that chain, starting at $t = 0$ with $\mathbf{v}_0$, the "input observation" for the RBM. Therefore, $(\mathbf{v}_k, \mathbf{h}_k)$ for $k \to \infty$ is a sample from the joint $P(\mathbf{v}, \mathbf{h})$. The log-likelihood of a value $\mathbf{v}_0$ under the model of the RBM is

$$\log P(\mathbf{v}_0) = \log \sum_{\mathbf{h}} P(\mathbf{v}_0, \mathbf{h}) = \log \sum_{\mathbf{h}} e^{-\text{energy}(\mathbf{v}_0, \mathbf{h})} - \log \sum_{\mathbf{v}, \mathbf{h}} e^{-\text{energy}(\mathbf{v}, \mathbf{h})}$$

and its gradient with respect to $\theta = (W, b, c)$ is

$$\frac{\partial \log P(\mathbf{v}_0)}{\partial \theta} = -\sum_{\mathbf{h}_0} Q(\mathbf{h}_0|\mathbf{v}_0) \frac{\partial \text{energy}(\mathbf{v}_0, \mathbf{h}_0)}{\partial \theta} + \sum_{\mathbf{v}_k, \mathbf{h}_k} P(\mathbf{v}_k, \mathbf{h}_k) \frac{\partial \text{energy}(\mathbf{v}_k, \mathbf{h}_k)}{\partial \theta}$$

for $k \to \infty$. An unbiased sample is $-\frac{\partial \text{energy}(\mathbf{v}_0, \mathbf{h}_0)}{\partial \theta} + E_{h_k}\left[\frac{\partial \text{energy}(\mathbf{v}_k, \mathbf{h}_k)}{\partial \theta}|v_k\right]$, where $\mathbf{h}_0$ is a sample from $Q(\mathbf{h}_0|\mathbf{v}_0)$ and $(\mathbf{v}_k, \mathbf{h}_k)$ is a sample of the Markov chain, and the expectation can be easily computed thanks to $P(h_k|v_k)$ factorizing. The idea of the Contrastive Divergence algorithm (Hinton, 2002) is to take $k$ small (typically $k = 1$). A pseudo-code for Contrastive Divergence training (with $k = 1$) of an RBM with binomial input and hidden units is presented in the Appendix (Algorithm `RBMupdate(x, ε, W, b, c)`). This procedure is called repeatedly with $\mathbf{v}_0 = x$ sampled from the training distribution for the RBM. To decide when to stop one may use a proxy for the training criterion, such as the reconstruction error $-\log P(\mathbf{v}_1 = x|\mathbf{v}_0 = x)$.

## 2.3 Greedy layer-wise training of a DBN

A greedy layer-wise training algorithm was proposed (Hinton et al., 2006) to train a DBN one layer at a time. One first trains an RBM that takes the empirical data as input and models it. Denote $Q(\mathbf{g}^1|\mathbf{g}^0)$ the posterior over $\mathbf{g}^1$ associated with that trained RBM (we recall that $\mathbf{g}^0 = x$ with $x$ the observed input). This gives rise to an "empirical" distribution $\widehat{p}^1$ over the first layer $\mathbf{g}^1$, when $\mathbf{g}^0$ is sampled from the data empirical distribution $\widehat{p}$: we have $\widehat{p}^1(\mathbf{g}^1) = \sum_{\mathbf{g}^0} \widehat{p}(\mathbf{g}^0)Q(\mathbf{g}^1|\mathbf{g}^0)$.

Note that a 1-level DBN is an RBM. The basic idea of the greedy layer-wise strategy is that after training the top-level RBM of a $\ell$-level DBN, one changes the interpretation of the RBM parameters to insert them in a $(\ell + 1)$-level DBN: the distribution $P(\mathbf{g}^{\ell-1}|\mathbf{g}^\ell)$ from the RBM associated with layers $\ell - 1$ and $\ell$ is kept as part of the DBN generative model. In the RBM between layers $\ell - 1$ and $\ell$, $P(g^\ell)$ is defined in terms on the parameters of that RBM, whereas in the DBN $P(g^\ell)$ is defined in terms of the parameters of the upper layers. Consequently, $Q(\mathbf{g}^\ell|\mathbf{g}^{\ell-1})$ of the RBM does not correspond to $P(\mathbf{g}^\ell|\mathbf{g}^{\ell-1})$ in the DBN, except when that RBM is the top layer of the DBN. However, we use $Q(\mathbf{g}^\ell|\mathbf{g}^{\ell-1})$ of the RBM as an approximation of the posterior $P(\mathbf{g}^\ell|\mathbf{g}^{\ell-1})$ for the DBN.

The samples of $\mathbf{g}^{\ell-1}$, with empirical distribution $\widehat{p}^{\ell-1}$, are converted stochastically into samples of $\mathbf{g}^\ell$ with distribution $\widehat{p}^\ell$ through $\widehat{p}^\ell(\mathbf{g}^\ell) = \sum_{\mathbf{g}^{\ell-1}} \widehat{p}^{\ell-1}(\mathbf{g}^{\ell-1})Q(\mathbf{g}^\ell|\mathbf{g}^{\ell-1})$. Although $\widehat{p}^\ell$ cannot be represented explicitly it is easy to sample unbiasedly from it: pick a training example and propagate it stochastically through the $Q(\mathbf{g}^i|\mathbf{g}^{i-1})$ at each level. As a nice side benefit, one obtains an approximation of the posterior for all the hidden variables in the DBN, at all levels, given an input $\mathbf{g}^0 = x$. Mean-field propagation (see below) gives a fast deterministic approximation of posteriors $P(\mathbf{g}^\ell|x)$.

Note that if we consider all the layers of a DBN from level $i$ to the top, we have a smaller DBN, which generates the marginal distribution $P(\mathbf{g}^i)$ for the complete DBN. The motivation for the greedy procedure is that a partial DBN with $\ell - i$ levels starting above level $i$ may provide a better model for $P(\mathbf{g}^i)$ than does the RBM initially associated with level $i$ itself.

The above greedy procedure is justified using a variational bound (Hinton et al., 2006). As a consequence of that bound, when inserting an additional layer, if it is initialized appropriately and has enough units, one can guarantee that initial improvements on the training criterion for the next layer

(fitting $\widehat{p}^{\ell}$) will yield improvement on the training criterion for the previous layer (likelihood with respect to $\widehat{p}^{\ell-1}$). The greedy layer-wise training algorithm for DBNs is quite simple, as illustrated by the pseudo-code in Algorithm `TrainUnsupervisedDBN` of the Appendix.

## 2.4 Supervised fine-tuning

As a last training stage, it is possible to fine-tune the parameters of all the layers together. For example Hinton et al. (2006) propose to use the wake-sleep algorithm (Hinton, Dayan, Frey, & Neal, 1995) to continue unsupervised training. Hinton et al. (2006) also propose to optionally use a mean-field approximation of the posteriors $P(\mathbf{g}^i|\mathbf{g}^0)$, by replacing the samples $\mathbf{g}_j^{i-1}$ at level $i-1$ by their bit-wise mean-field expected value $\mu_j^{i-1}$, with $\mu^i = \text{sigm}(b^i + W^i \mu^{i-1})$. According to these propagation rules, the whole network now deterministically computes internal representations as functions of the network input $\mathbf{g}^0 = x$. After unsupervised pre-training of the layers of a DBN following Algorithm `TrainUnsupervisedDBN` (see Appendix) the whole network can be further optimized by gradient descent with respect to any deterministically computable training criterion that depends on these representations. For example, this can be used (Hinton & Salakhutdinov, 2006) to fine-tune a very deep auto-encoder, minimizing a reconstruction error. It is also possible to use this as initialization of all except the last layer of a traditional multi-layer neural network, using gradient descent to fine-tune the whole network with respect to a supervised training criterion.

Algorithm `DBNSupervisedFineTuning` in the appendix contains pseudo-code for supervised fine-tuning, as part of the global supervised learning algorithm `TrainSupervisedDBN`. Note that better results were obtained when using a *20-fold larger learning rate* with the supervised criterion (here, squared error or cross-entropy) updates than in the contrastive divergence updates.

## 3  Extension to continuous-valued inputs

With the binary units introduced for RBMs and DBNs in Hinton et al. (2006) one can "cheat" and handle continuous-valued inputs by scaling them to the (0,1) interval and considering each input continuous value as the probability for a binary random variable to take the value 1. This has worked well for pixel gray levels, but it may be inappropriate for other kinds of input variables. Previous work on continuous-valued input in RBMs include (Chen & Murray, 2003), in which noise is added to sigmoidal units, and the RBM forms a special form of Diffusion Network (Movellan, Mineiro, & Williams, 2002). We concentrate here on simple extensions of the RBM framework in which only the energy function and the allowed range of values are changed.

**Linear energy: exponential or truncated exponential**

Consider a unit with value $y$ of an RBM, connected to units $\mathbf{z}$ of the other layer. $p(y|\mathbf{z})$ can be obtained from the terms in the exponential that contain $y$, which can be grouped in $ya(\mathbf{z})$ for linear energy functions as in (2), where $a(\mathbf{z}) = b + w'\mathbf{z}$ with $b$ the bias of unit $y$, and $w$ the vector of weights connecting unit $y$ to units $\mathbf{z}$. If we allow $y$ to take any value in interval $I$, the conditional density of $y$ becomes $p(y|\mathbf{z}) = \dfrac{exp(ya(\mathbf{z}))\mathbf{1}_{y\in I}}{\int_v exp(va(\mathbf{z}))\mathbf{1}_{v\in I}dv}$. When $I = [0,\infty)$, this is an exponential density with parameter $a(\mathbf{z})$, and the normalizing integral equals $-1/a(\mathbf{z})$, but only exists if $\forall \mathbf{z}, a(\mathbf{z}) < 0$ Computing the density, computing the expected value $(= -1/a(\mathbf{z}))$ and sampling would all be easy.

Alternatively, if $I$ is a closed interval (as in many applications of interest), or if we would like to use such a unit as a hidden unit with *non-linear expected value*, the above density is a *truncated exponential*. For simplicity we consider the case $I = [0,1]$ here, for which the normalizing integral, which always exists, is $\frac{exp(-a(\mathbf{z}))-1}{a(\mathbf{z})}$. The conditional expectation of $u$ given $\mathbf{z}$ is interesting because it has a sigmoidal-like saturating and monotone non-linearity: $E[y|\mathbf{z}] = \frac{1}{1-exp(-a(\mathbf{z}))} - \frac{1}{a(\mathbf{z})}$. A sampling from the truncated exponential is easily obtained from a uniform sample $U$, using the inverse cumulative $F^{-1}$ of the conditional density $y|\mathbf{z}$: $F^{-1}(U) = \frac{\log(1-U\times(1-exp(a(\mathbf{z}))))}{a(\mathbf{z})}$. In both truncated and not truncated cases, the Contrastive Divergence updates have the same form as for binomial units (input value times output value), since the updates only depend on the derivative of the energy with respect to the parameters. Only sampling is changed, according to the unit's conditional density.

**Quadratic energy: Gaussian units**

To obtain Gaussian-distributed units, one adds quadratic terms to the energy. Adding $\sum_i d_i^2 y_i^2$ gives rise to a diagonal covariance matrix between units of the same layer, where $y_i$ is the continuous value of a Gaussian unit and $d_i^2$ is a positive parameter that is equal to the inverse of the variance of $y_i$. In

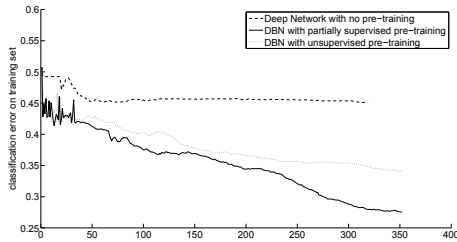

Figure 1: Training classification error vs training iteration, on the Cotton price task, for deep network without pre-training, for DBN with unsupervised pre-training, and DBN with partially supervised pre-training. Illustrates optimization difficulty of deep networks and advantage of partially supervised training.

| | Abalone | | | Cotton | | |
|---|---|---|---|---|---|---|
| | train. | valid. | test. | train. | valid. | test. |
| 1. Deep Network with no pre-training | 4.23 | 4.43 | 4.2 | 45.2% | 42.9% | 43.0% |
| 2. Logistic regression | · | · | · | 44.0% | 42.6% | 45.0% |
| 3. DBN, binomial inputs, unsupervised | 4.59 | 4.60 | 4.47 | 44.0% | 42.6% | 45.0% |
| 4. DBN, binomial inputs, partially supervised | 4.39 | 4.45 | 4.28 | 43.3% | 41.1% | 43.7% |
| 5. DBN, Gaussian inputs, unsupervised | 4.25 | 4.42 | 4.19 | 35.7% | 34.9% | 35.8% |
| 6. DBN, Gaussian inputs, partially supervised | 4.23 | 4.43 | 4.18 | 27.5% | 28.4% | 31.4% |

Table 1: *Mean squared prediction error on Abalone task and classification error on Cotton task, showing improvement with Gaussian units.*

this case the variance is unconditional, whereas the mean depends on the inputs of the unit: for a unit $y$ with inputs $\mathbf{z}$ and inverse variance $d^2$, $E[y|\mathbf{z}] = \frac{a(\mathbf{z})}{2d^2}$.

The Contrastive Divergence updates are easily obtained by computing the derivative of the energy with respect to the parameters. For the parameters in the linear terms of the energy function (e.g., $b$ and $w$ above), the derivatives have the same form (input unit value times output unit value) as for the case of binomial units. For quadratic parameter $d > 0$, the derivative is simply $2dy^2$. Gaussian units were previously used as hidden units of an RBM (with binomial or multinomial inputs) applied to an information retrieval task (Welling, Rosen-Zvi, & Hinton, 2005). Our interest here is to use them for continuous-valued inputs.

**Using continuous-valued hidden units**

Although we have introduced RBM units with continuous values to better deal with the representation of input variables, they could also be considered for use in the hidden layers, in replacement or complementing the binomial units which have been used in the past. However, Gaussian and exponential hidden units have a weakness: the mean-field propagation through a Gaussian unit gives rise to a purely linear transformation. Hence if we have only such linear hidden units in a multi-layered network, the mean-field propagation function that maps inputs to internal representations would be completely linear. In addition, in a DBN containing only Gaussian units, one would only be able to model Gaussian data. On the other hand, combining Gaussian with other types of units could be interesting. In contrast with Gaussian or exponential units, remark that the conditional expectation of **truncated** exponential units is non-linear, and in fact involves a sigmoidal form of non-linearity applied to the weighted sum of its inputs.

**Experiment 1**

This experiment was performed on two data sets: the UCI repository Abalone data set (split in 2177 training examples, 1000 validation examples, 1000 test examples) and a financial data set. The latter has real-valued input variables representing averages of returns and squared returns for which the binomial approximation would seem inappropriate. The target variable is next month's return of a Cotton futures contract. There are 13 continuous input variables, that are averages of returns over different time-windows up to 504 days. There are 3135 training examples, 1000 validation examples, and 1000 test examples. The dataset is publicly available at `http://www.iro.umontreal.ca/˜lisa/fin_data/`. In Table 1 (rows 3 and 5), we show improvements brought by DBNs with Gaussian inputs over DBNs with binomial inputs (with binomial hidden units in both cases). The networks have two hidden layers. All hyper-parameters are selected based on validation set performance.

# 4   Understanding why the layer-wise strategy works

A reasonable explanation for the apparent success of the layer-wise training strategy for DBNs is that unsupervised pre-training helps to mitigate the difficult optimization problem of deep networks by better initializing the weights of all layers. Here we present experiments that support and clarify this.

**Training each layer as an auto-encoder**

We want to verify that the layer-wise greedy unsupervised pre-training principle can be applied when using an auto-encoder instead of the RBM as a layer building block. Let $x$ be the input vector with $x_i \in (0,1)$. For a layer with weights matrix $W$, hidden biases column vector $b$ and input biases column vector $c$, the reconstruction probability for bit $i$ is $p_i(x)$, with the vector of probabilities $p(x) = \text{sigm}(c + W\text{sigm}(b + W'x))$. The training criterion for the layer is the average of negative log-likelihoods for predicting $x$ from $p(x)$. For example, if $x$ is interpreted either as a sequence of bits or a sequence of bit probabilities, we minimize the reconstruction cross-entropy: $R = -\sum_i x_i \log p_i(x) + (1 - x_i)\log(1 - p_i(x))$. We report several experimental results using this training criterion for each layer, in comparison to the contrastive divergence algorithm for an RBM. Pseudo-code for a deep network obtained by training each layer as an auto-encoder is given in Appendix (Algorithm `TrainGreedyAutoEncodingDeepNet`).

One question that arises with auto-encoders in comparison with RBMs is whether the auto-encoders will fail to learn a useful representation when the number of units is not strictly decreasing from one layer to the next (since the networks could theoretically just learn to be the identity and perfectly minimize the reconstruction error). However, our experiments suggest that networks with non-decreasing layer sizes generalize well. This might be due to weight decay and stochastic gradient descent, preventing large weights: optimization falls in a local minimum which corresponds to a good transformation of the input (that provides a good initialization for supervised training of the whole net).

**Greedy layer-wise supervised training**

A reasonable question to ask is whether the fact that each layer is trained in an unsupervised way is critical or not. An alternative algorithm is supervised, greedy and layer-wise: train each new hidden layer as the hidden layer of a one-hidden layer supervised neural network *NN* (taking as input the output of the last of previously trained layers), and then throw away the output layer of *NN* and use the parameters of the hidden layer of *NN* as pre-training initialization of the new top layer of the deep net, to map the output of the previous layers to a hopefully better representation. Pseudo-code for a deep network obtained by training each layer as the hidden layer of a supervised one-hidden-layer neural network is given in Appendix (Algorithm `TrainGreedySupervisedDeepNet`).

**Experiment 2**.

We compared the performance on the MNIST digit classification task obtained with five algorithms: (a) DBN, (b) deep network whose layers are initialized as auto-encoders, (c) above described supervised greedy layer-wise algorithm to pre-train each layer, (d) deep network with no pre-training (random initialization), (e) shallow network (1 hidden layer) with no pre-training.

The final fine-tuning is done by adding a logistic regression layer on top of the network and training the whole network by stochastic gradient descent on the cross-entropy with respect to the target classification. The networks have the following architecture: 784 inputs, 10 outputs, 3 hidden layers with variable number of hidden units, selected by validation set performance (typically selected layer sizes are between 500 and 1000). The shallow network has a single hidden layer. An L2 weight decay hyper-parameter is also optimized. The DBN was slower to train and less experiments were performed, so that longer training and more appropriately chosen sizes of layers and learning rates could yield better results (Hinton 2006, unpublished, reports 1.15% error on the MNIST test set).

|  | Experiment 2 | | | Experiment 3 | | |
|---|---|---|---|---|---|---|
|  | train. | valid. | test | train. | valid. | test |
| DBN, unsupervised pre-training | 0% | 1.2% | 1.2% | 0% | 1.5% | 1.5% |
| Deep net, auto-associator pre-training | 0% | 1.4% | 1.4% | 0% | 1.4% | 1.6% |
| Deep net, supervised pre-training | 0% | 1.7% | 2.0% | 0% | 1.8% | 1.9% |
| Deep net, no pre-training | .004% | 2.1% | 2.4% | .59% | 2.1% | 2.2% |
| Shallow net, no pre-training | .004% | 1.8% | 1.9% | 3.6% | 4.7% | 5.0% |

Table 2: *Classification error on MNIST training, validation, and test sets, with the best hyperparameters according to validation error, with and without pre-training, using purely supervised or purely unsupervised pre-training. In experiment 3, the size of the top hidden layer was set to 20.*

On MNIST, differences of more than .1% are statistically significant. The results in Table 2 suggest that the auto-encoding criterion can yield performance comparable to the DBN when the layers are finally tuned in a supervised fashion. They also clearly show that the greedy unsupervised layer-wise pre-training gives much better results than the standard way to train a deep network (with no greedy

pre-training) or a shallow network, and that, without pre-training, deep networks tend to perform worse than shallow networks. The results also suggest that *unsupervised greedy layer-wise pre-training can perform significantly better than purely supervised greedy layer-wise pre-training*. A possible explanation is that the greedy supervised procedure is **too greedy**: in the learned hidden units representation it may discard some of the information about the target, information that cannot be captured easily by a one-hidden-layer neural network but could be captured by composing more hidden layers.

**Experiment 3**

However, there is something troubling in the Experiment 2 results (Table 2): all the networks, even those without greedy layer-wise pre-training, perform almost perfectly on the *training set*, which would appear to contradict the hypothesis that the main effect of the layer-wise greedy strategy is to help the optimization (with poor optimization one would expect poor training error). A possible explanation coherent with our initial hypothesis and with the above results is captured by the following **hypothesis**. Without pre-training, the lower layers are initialized poorly, but still allowing the top two layers to learn the training set almost perfectly, because the output layer and the last hidden layer form a standard shallow but fat neural network. Consider the top two layers of the deep network *with pre-training*: it presumably takes as input a *better representation*, one that allows for better generalization. Instead, the network *without pre-training* sees a "random" transformation of the input, one that preserves enough information about the input to fit the training set, but that does not help to generalize. To test that hypothesis, we performed a second series of experiments in which we constrain the top hidden layer to be small (20 hidden units). The Experiment 3 results (Table 2) clearly confirm our hypothesis. With no pre-training, training error degrades significantly when there are only 20 hidden units in the top hidden layer. In addition, the results obtained without pre-training were found to have extremely large variance indicating high sensitivity to initial conditions. Overall, the results in the tables and in Figure 1 are consistent with the hypothesis that the greedy layer-wise procedure essentially helps to better optimize the deep networks, *probably by initializing the hidden layers so that they represent more meaningful representations of the input*, which also yields to better generalization.

**Continuous training of all layers of a DBN**

With the layer-wise training algorithm for DBNs (`TrainUnsupervisedDBN` in Appendix), one element that we would like to dispense with is having to decide the number of training iterations for each layer. It would be good if we did not have to explicitly add layers one at a time, i.e., if we could train all layers simultaneously, but keeping the "greedy" idea that *each layer is pre-trained to model its input, ignoring the effect of higher layers*. To achieve this it is sufficient to insert a line in `TrainUnsupervisedDBN`, so that `RBMupdate` is called on all the layers and the stochastic hidden values are propagated all the way up. Experiments with this variant demonstrated that it works at least as well as the original algorithm. The advantage is that we can now have a single stopping criterion (for the whole network). Computation time is slightly greater, since we do more computations initially (on the upper layers), which might be wasted (before the lower layers converge to a decent representation), but time is saved on optimizing hyper-parameters. This variant may be more appealing for on-line training on very large data-sets, where one would never cycle back on the training data.

## 5   Dealing with uncooperative input distributions

In classification problems such as MNIST where classes are well separated, the structure of the input distribution $p(x)$ naturally contains much information about the target variable $y$. Imagine a supervised learning task in which the input distribution is mostly unrelated with $y$. In regression problems, which we are interested in studying here, this problem could be much more prevalent. For example imagine a task in which $x \sim p(x)$ and the target $y = f(x) +$ noise (e.g., $p$ is Gaussian and $f =$ sinus) *with no particular relation* between $p$ and $f$. In such settings we cannot expect the unsupervised greedy layer-wise pre-training procedure to help in training deep supervised networks. To deal with such uncooperative input distributions, we propose to train each layer with a mixed training criterion that combines the unsupervised objective (modeling or reconstructing the input) and a supervised objective (helping to predict the target). A simple algorithm thus adds the updates on the hidden layer weights from the unsupervised algorithm (Contrastive Divergence or reconstruction error gradient) with the updates from the gradient on a supervised prediction error, using a temporary output layer, as with the greedy layer-wise supervised training algorithm. In our experiments it appeared sufficient to perform that partial supervision with the **first layer only**, since once the predictive information about the target is "forced" into the representation of the first layer, it tends to stay in the upper layers. The results in Figure 1 and Table 1 clearly show the advantage of this *partially supervised greedy training*

*algorithm*, in the case of the financial dataset. Pseudo-code for partially supervising the first (or later layer) is given in Algorithm `TrainPartiallySupervisedLayer` (in the Appendix).

# 6 Conclusion

This paper is motivated by the need to develop good training algorithms for deep architectures, since these can be much more representationally efficient than shallow ones such as SVMs and one-hidden-layer neural nets. We study Deep Belief Networks applied to supervised learning tasks, and the principles that could explain the good performance they have yielded. The three principal contributions of this paper are the following. First we extended RBMs and DBNs in new ways to naturally handle continuous-valued inputs, showing examples where much better predictive models can thus be obtained. Second, we performed experiments which support the hypothesis that the greedy unsupervised layer-wise training strategy helps to *optimize deep networks*, but suggest that better generalization is also obtained because this strategy initializes upper layers with *better representations* of relevant *high-level abstractions*. These experiments suggest a general principle that can be applied beyond DBNs, and we obtained similar results when each layer is initialized as an auto-associator instead of as an RBM. Finally, although we found that it is important to have an unsupervised component to train each layer (a fully supervised greedy layer-wise strategy performed worse), we studied supervised tasks in which the structure of the input distribution is not revealing enough of the conditional density of $y$ given $x$. In that case the DBN unsupervised greedy layer-wise strategy appears inadequate and we proposed a simple fix based on partial supervision, that can yield significant improvements.

# References

Allender, E. (1996). Circuit complexity before the dawn of the new millennium. In *16th Annual Conference on Foundations of Software Technology and Theoretical Computer Science*, pp. 1–18. Lecture Notes in Computer Science 1180.

Bengio, Y., Delalleau, O., & Le Roux, N. (2006). The curse of highly variable functions for local kernel machines. In Weiss, Y., Schölkopf, B., & Platt, J. (Eds.), *Advances in Neural Information Processing Systems 18*, pp. 107–114. MIT Press, Cambridge, MA.

Bengio, Y., & Le Cun, Y. (2007). Scaling learning algorithms towards AI. In Bottou, L., Chapelle, O., DeCoste, D., & Weston, J. (Eds.), *Large Scale Kernel Machines*. MIT Press.

Bengio, Y., Le Roux, N., Vincent, P., Delalleau, O., & Marcotte, P. (2006). Convex neural networks. In Weiss, Y., Schölkopf, B., & Platt, J. (Eds.), *Advances in Neural Information Processing Systems 18*, pp. 123–130. MIT Press, Cambridge, MA.

Chen, H., & Murray, A. (2003). A continuous restricted boltzmann machine with an implementable training algorithm. *IEE Proceedings of Vision, Image and Signal Processing*, *150*(3), 153–158.

Fahlman, S., & Lebiere, C. (1990). The cascade-correlation learning architecture. In Touretzky, D. (Ed.), *Advances in Neural Information Processing Systems 2*, pp. 524–532 Denver, CO. Morgan Kaufmann, San Mateo.

Hastad, J. T. (1987). *Computational Limitations for Small Depth Circuits*. MIT Press, Cambridge, MA.

Hinton, G. E., Osindero, S., & Teh, Y. (2006). A fast learning algorithm for deep belief nets. *Neural Computation*, *18*, 1527–1554.

Hinton, G. (2002). Training products of experts by minimizing contrastive divergence. *Neural Computation*, *14*(8), 1771–1800.

Hinton, G., Dayan, P., Frey, B., & Neal, R. (1995). The wake-sleep algorithm for unsupervised neural networks. *Science*, *268*, 1558–1161.

Hinton, G., & Salakhutdinov, R. (2006). Reducing the dimensionality of data with neural networks. *Science*, *313*(5786), 504–507.

Lengellé, R., & Denoeux, T. (1996). Training MLPs layer by layer using an objective function for internal representations. *Neural Networks*, *9*, 83–97.

Movellan, J., Mineiro, P., & Williams, R. (2002). A monte-carlo EM approach for partially observable diffusion processes: theory and applications to neural networks. *Neural Computation*, *14*, 1501–1544.

Tesauro, G. (1992). Practical issues in temporal difference learning. *Machine Learning*, *8*, 257–277.

Utgoff, P., & Stracuzzi, D. (2002). Many-layered learning. *Neural Computation*, *14*, 2497–2539.

Welling, M., Rosen-Zvi, M., & Hinton, G. E. (2005). Exponential family harmoniums with an application to information retrieval. In *Advances in Neural Information Processing Systems*, Vol. 17 Cambridge, MA. MIT Press.
